# Neural Learning in Structured Parameter Spaces — Natural Riemannian Gradient

**Shun-ichi Amari**
RIKEN Frontier Research Program, RIKEN,
Hirosawa 2-1, Wako-shi 351-01, Japan
amari@zoo.riken.go.jp

## Abstract

The parameter space of neural networks has a Riemannian metric structure. The natural Riemannian gradient should be used instead of the conventional gradient, since the former denotes the true steepest descent direction of a loss function in the Riemannian space. The behavior of the stochastic gradient learning algorithm is much more effective if the natural gradient is used. The present paper studies the information-geometrical structure of perceptrons and other networks, and prove that the on-line learning method based on the natural gradient is asymptotically as efficient as the optimal batch algorithm. Adaptive modification of the learning constant is proposed and analyzed in terms of the Riemannian measure and is shown to be efficient. The natural gradient is finally applied to blind separation of mixtured independent signal sources.

## 1 Introduction

Neural learning takes place in the parameter space of modifiable synaptic weights of a neural network. The role of each parameter is different in the neural network so that the parameter space is structured in this sense. The Riemannian structure which represents a local distance measure is introduced in the parameter space by information geometry (Amari, 1985).

On-line learning is mostly based on the stochastic gradient descent method, where the current weight vector is modified in the gradient direction of a loss function. However, the ordinary gradient does not represent the steepest direction of a loss function in the Riemannian space. A geometrical modification is necessary, and it is called the natural Riemannian gradient. The present paper studies the remarkable effects of using the natural Riemannian gradient in neural learning.

We first studies the asymptotic behavior of on-line learning (Opper, NIPS'95 Workshop). Batch learning uses all the examples at any time to obtain the optimal weight vector, whereas on-line learning uses an example once when it is observed. Hence, in general, the target weight vector is estimated more accurately in the case of batch learning. However, we prove that, when the Riemannian gradient is used, on-line learning is asymptotically as efficient as optimal batch learning.

On-line learning is useful when the target vector fluctuates slowly (Amari, 1967). In this case, we need to modify a learning constant $\eta_t$ depending on how far the current weight vector is located from the target function. We show an algorithm adaptive changes in the learning constant based on the Riemannian criterion and prove that it gives asymptotically optimal behavior. This is a generalization of the idea of Sompolinsky et al. [1995].

We then answer the question what is the Riemannian structure to be introduced in the parameter space of synaptic weights. We answer this problem from the point of view of information geometry (Amari [1985, 1995], Amari et al [1992]). The explicit form of the Riemannian metric and its inverse matrix are given in the case of simple perceptrons.

We finally show how the Riemannian gradient is applied to blind separation of mixtured independent signal sources. Here, the mixing matrix is unknown so that the parameter space is the space of matrices. The Riemannian structure is introduced in it. The natural Riemannian gradient is computationary much simpler and more effective than the conventional gradient.

## 2 Stochastic Gradient Descent and On-Line Learning

Let us consider a neural network which is specified by a vector parameter $w = (w_1, \cdots w_n) \in R^n$. The parameter $w$ is composed of modifiable connection weights and thresholds. Let us denote by $l(x, w)$ a loss when input signal $x$ is processed by a network having parameter $w$. In the case of multilayer perceptrons, a desired output or teacher signal $y$ is associated with $x$, and a typical loss is given

$$l(x, y, w) = \frac{1}{2} \parallel y - f(x, w) \parallel^2, \tag{1}$$

where $z = f(x, w)$ is the output from the network.

When input $x$, or input-output training pair $(x, y)$, is generated from a fixed probability distribution, the expected loss $L(w)$ of the network specified by $w$ is

$$L(w) = \mathrm{E}[l(x, y; w)], \tag{2}$$

where E denotes the expectation. A neural network is trained by using training examples $(x_1, y_1), (x_2, y_2), \cdots$ to obtain the optimal network parameter $w^*$ that minimizes $L(w)$. If $L(w)$ is known, the gradient method is described by

$$w_{t+1} = w_t - \eta_t \nabla L(w_t), \qquad t = 1, 2, \cdots$$

where $\eta_t$ is a learning constant depending on $t$ and $\nabla L = \partial L / \partial w$. Usually $L(w)$ is unknown. The stochastic gradient learning method

$$w_{t+1} = w_t - \eta_t \nabla l(x_{t+1}, y_{t+1}; w_t) \tag{3}$$

was proposed by an old paper (Amari [1967]). This method has become popular since Rumelhart et al. [1986] rediscovered it. It is expected that, when $\eta_t$ converges to 0 in a certain manner, the above $w_t$ converges to $w^*$. The dynamical behavior of

(3) was studied by Amari [1967], Heskes and Kappen [1992] and many others when $\eta_t$ is a constant.

It was also shown in Amari [1967] that

$$\boldsymbol{w}_{t+1} = \boldsymbol{w}_t - \eta_t G^{-1} \nabla l(\boldsymbol{x}_{t+1}, \boldsymbol{y}_{t+1}, \boldsymbol{w}_t) \tag{4}$$

works well for any positive-definite matrix, in particular for the metric $G$. Geometrically speaking $\partial l / \partial \boldsymbol{w}$ is a covariant vector while $\Delta \boldsymbol{w}_t = \boldsymbol{w}_{t+1} - \boldsymbol{w}_t$ is a contravariant vector. Therefore, it is natural to use a (contravariant) metric tensor $G^{-1}$ to convert the covariant gradient into the contravariant form

$$\tilde{\nabla} l = G^{-1} \frac{\partial l}{\partial \boldsymbol{w}} = \left( \sum_j g^{ij} \frac{\partial}{\partial w_j}(\boldsymbol{w}) \right), \tag{5}$$

where $G^{-1} = (g^{ij})$ is the inverse matrix of $G = (g_{ij})$. The present paper studies how the matrix tensor matrix $G$ should be defined in neural learning and how effective is the new gradient learning rule

$$\boldsymbol{w}_{t+1} = \boldsymbol{w}_t - \eta_t \tilde{\nabla} l(\boldsymbol{x}_t, \boldsymbol{y}_t, \boldsymbol{w}_t). \tag{6}$$

## 3   Gradient in Riemannian spaces

Let $S = \{\boldsymbol{w}\}$ be the parameter space and let $l(\boldsymbol{w})$ be a function defined on $S$. When $S$ is a Euclidean space and $\boldsymbol{w}$ is an orthonormal coordinate system, the squared length of a small incremental vector $d\boldsymbol{w}$ connecting $\boldsymbol{w}$ and $\boldsymbol{w} + d\boldsymbol{w}$ is given by

$$|d\boldsymbol{w}|^2 = \sum_{i=1}^{n} (dw_i)^2. \tag{7}$$

However, when the coordinate system is non-orthonormal or the space $S$ is Riemannian, the squared length is given by a quadratic form

$$|d\boldsymbol{w}|^2 = \sum_{i,j} g_{ij}(\boldsymbol{w}) dw_i dw_j = \boldsymbol{w}' G \boldsymbol{w}. \tag{8}$$

Here, the matrix $G = (g_{ij})$ depends in general on $\boldsymbol{w}$ and is called the metric tensor. It reduces to the identity matrix $I$ in the Euclidean orthonormal case. It will be shown soon that the parameter space $S$ of neural networks has Riemannian structure (see Amari et al. [1992], Amari [1995], etc.).

The steepest descent direction of a function $l(\boldsymbol{w})$ at $\boldsymbol{w}$ is defined by a vector $d\boldsymbol{w}$ that minimize $l(\boldsymbol{w} + d\boldsymbol{w})$ under the constraint $|d\boldsymbol{w}|^2 = \varepsilon^2$ (see eq.8) for a sufficiently small constant $\varepsilon$.

**Lemma 1.** The steepest descent direction of $l(\boldsymbol{w})$ in a Riemannian space is given by

$$-\tilde{\nabla} l(\boldsymbol{w}) = -G^{-1}(\boldsymbol{w}) \nabla l(\boldsymbol{w}).$$

We call

$$\tilde{\nabla} l(\boldsymbol{w}) = G^{-1}(\boldsymbol{w}) \nabla l(\boldsymbol{w})$$

the natural gradient of $l(\boldsymbol{w})$ in the Riemannian space. It shows the steepest descent direction of $l$, and is nothing but the contravariant form of $\nabla l$ in the tensor notation. When the space is Euclidean and the coordinate system is orthonormal, $G$ is the unit matrix $I$ so that $\tilde{\nabla} l = \nabla l$.

## 4  Natural gradient gives efficient on-line learning

Let us begin with the simplest case of noisy multilayer analog perceptrons. Given input $\boldsymbol{x}$, the network emits output $\boldsymbol{z} = \boldsymbol{f}(\boldsymbol{x}, \boldsymbol{w}) + \boldsymbol{n}$, where $\boldsymbol{f}$ is a differentiable deterministic function of the multilayer perceptron with parameter $\boldsymbol{w}$ and $\boldsymbol{n}$ is a noise subject to the normal distribution $N(0, I)$. The probability density of an input-output pair $(\boldsymbol{x}, \boldsymbol{z})$ is given by

$$p(\boldsymbol{x}, \boldsymbol{z}; \boldsymbol{w}) = q(\boldsymbol{x}) p(\boldsymbol{z}|\boldsymbol{x}; \boldsymbol{w}),$$

where $q(\boldsymbol{x})$ is the probability distribution of input $\boldsymbol{x}$, and

$$p(\boldsymbol{z}|\boldsymbol{x}; \boldsymbol{w}) = \frac{1}{\sqrt{2\pi}} \exp\left\{ -\frac{1}{2} \parallel \boldsymbol{z} - \boldsymbol{f}(\boldsymbol{x}, \boldsymbol{w}) \parallel^2 \right\}.$$

The squared error loss function (1) can be written as

$$l(\boldsymbol{x}, \boldsymbol{z}, \boldsymbol{w}) = -\log p(\boldsymbol{x}, \boldsymbol{z}; \boldsymbol{w}) + \log q(\boldsymbol{x}) - \log \sqrt{2\pi}.$$

Hence, minimizing the loss is equivalent to maximizing the likelihood function $p(\boldsymbol{x}, \boldsymbol{z}; \boldsymbol{w})$.

Let $D_T = \{(\boldsymbol{x}_1, \boldsymbol{z}_1), \cdots, (\boldsymbol{x}_T, \boldsymbol{z}_T)\}$ be $T$ independent input-output examples generated by the network having the parameter $\boldsymbol{w}^*$. Then, maximum likelihood estimator $\hat{\boldsymbol{w}}_T$ minimizes the log loss $l(\boldsymbol{x}, \boldsymbol{z}; \boldsymbol{w}) = -\log p(\boldsymbol{x}, \boldsymbol{z}; \boldsymbol{w})$ over the training data $D_T$, that is, it minimizes the training error

$$E_{\text{train}}(\boldsymbol{w}) = \frac{1}{T} \sum_{t=1}^{T} l(\boldsymbol{x}_t, \boldsymbol{z}_t; \boldsymbol{w}). \tag{9}$$

The maximum likelihood estimator is efficient (or Fisher-efficient), implying that it is the best consistent estimator satisfying the Cramér-Rao bound asymptotically,

$$\lim_{T \to \infty} T \mathrm{E}[(\hat{\boldsymbol{w}}_T - \boldsymbol{w}^*)(\hat{\boldsymbol{w}}_T - \boldsymbol{w}^*)'] = G^{-1}, \tag{10}$$

where $G^{-1}$ is the inverse of the Fisher information matrix $G = (g_{ij})$ defined by

$$g_{ij} = \mathrm{E}\left[ \frac{\partial \log p(\boldsymbol{x}, \boldsymbol{z}; \boldsymbol{w})}{\partial w_i} \frac{\partial \log p(\boldsymbol{x}, \boldsymbol{z}; \boldsymbol{w})}{\partial w_j} \right] \tag{11}$$

in the component form. Information geometry (Amari, 1985) proves that the Fisher information $G$ is the only invariant metric to be introduced in the space $S = \{\boldsymbol{w}\}$ of the parameters of probability distributions.

Examples $(\boldsymbol{x}_1, \boldsymbol{z}_1), (\boldsymbol{x}_2, \boldsymbol{z}_2) \cdots$ are given one at a time in the case of on-line learning. Let $\tilde{\boldsymbol{w}}_t$ be the estimated value at time $t$. At the next time $t+1$, the estimator $\tilde{\boldsymbol{w}}_t$ is modified to give a new estimator $\tilde{\boldsymbol{w}}_{t+1}$ based on the observation $(\boldsymbol{x}_{t+1}, \boldsymbol{z}_{t+1})$. The old observations $(\boldsymbol{x}_1, \boldsymbol{z}_1), \cdots, (\boldsymbol{x}_t, \boldsymbol{z}_t)$ cannot be reused to obtain $\tilde{\boldsymbol{w}}_{t+1}$, so that the learning rule is written as $\tilde{\boldsymbol{w}}_{t+1} = \boldsymbol{m}(\boldsymbol{x}_{t+1}, \boldsymbol{z}_{t+1}, \tilde{\boldsymbol{w}}_t)$. The process $\{\tilde{\boldsymbol{w}}_t\}$ is hence Markovian. Whatever a learning rule $\boldsymbol{m}$ we choose, the behavior of the estimator $\tilde{\boldsymbol{w}}_t$ is never better than that of the optimal batch estimator $\hat{\boldsymbol{w}}_t$ because of this restriction. The conventional on-line learning rule is given by the following gradient form $\tilde{\boldsymbol{w}}_{t+1} = \tilde{\boldsymbol{w}}_t - \eta_t \nabla l(\boldsymbol{x}_{t+1}, \boldsymbol{z}_{t+1}; \tilde{\boldsymbol{w}}_t)$. When $\eta_t$ satisfies a certain condition, say $\eta_t = c/t$, the stochastic approximation guarantees that $\tilde{\boldsymbol{w}}_t$ is a consistent estimator converging to $\boldsymbol{w}^*$. However, it is not efficient in general.

There arises a question if there exists an on-line learning rule that gives an efficient estimator. If it exists, the asymptotic behavior of on-line learning is equivalent to

that of the batch estimation method. The present paper answers the question by giving an efficient on-line learning rule

$$\tilde{w}_{t+1} = \tilde{w}_t - \frac{1}{t}\tilde{\nabla}l(x_{t+1}, z_{t+1}; \tilde{w}_t). \tag{12}$$

**Theorem 1.** The natural gradient on-line learning rule gives an Fisher-efficient estimator, that is,

$$\tilde{V}_t = E[(\tilde{w}_t - w^*)(\tilde{w}_t - w^*)'] \approx \frac{1}{t}G^{-1}(w^*). \tag{13}$$

## 5  Adaptive modification of learning constant

We have proved that $\eta_t = 1/t$ with the coefficient matrix $G^{-1}$ is the asymptotically best choice for on-line learning. However, when the target parameter $w^*$ is not fixed but fluctuating or changes suddenly, this choice is not good, because the learning system cannot follow the change if $\eta_t$ is too small. It was proposed in Amari [1967] to choose $\eta_t$ adaptively such that $\eta_t$ becomes larger when the current target $w^*$ is far from $w_t$ and becomes small when it is close to $w_t$ adaptively. However, no definite scheme was analyzed there. Sompolinsky et al. [1995] proposed an excellent scheme of an adaptive choice of $\eta_t$ for a deterministic dichotomy neural networks. We extend their idea to be applicable to stochastic cases, where the Riemannian structure plays a role.

We assume that $l(x, z; w)$ is differentiable with respect to $w$. (The non-differentiable case is usually more difficult to analyze. Sompolinsky et al [1995] treated this case.) We moreover treat the realizable teacher so that $L(w^*) = 0$.

We propose the following learning scheme:

$$\hat{w}_{t+1} = \hat{w}_t - \eta_t\tilde{\nabla}l(x_{t+1}, z_{t+1}; \hat{w}_t) \tag{14}$$
$$\eta_{t+1} = \eta_t + \alpha\eta_t[\beta l(x_{t+1}, z_{t+1}; \hat{w}_t) - \eta_t], \tag{15}$$

where $\alpha$ and $\beta$ are constants. We try to analyze the dynamical behavior of learning by using the continuous version of the algorithm,

$$\frac{d}{dt}\hat{w}_t = -\eta_t\tilde{\nabla}l(x_t, z_t; \hat{w}_t), \tag{16}$$
$$\frac{d}{dt}\eta_t = \alpha\eta_t[\beta l(x_t, z_t; \hat{w}_t) - \eta_t]. \tag{17}$$

In order to show the dynamical behavior of $(\hat{w}_t, \eta_t)$, we use the averaged version of the above equation with respect to the current input-output pair $(x_t, z_t)$. We introduce the squared error variable

$$e_t = \frac{1}{2}(w_t - w^*)'G^*(w_t - w^*). \tag{18}$$

By using the average and continuous time version

$$\dot{w}_t = -\eta_t G^{-1}(w_t)\left\langle\frac{\partial}{\partial w}l(x_t, z_t; w_t)\right\rangle,$$
$$\dot{\eta}_t = \alpha\eta_t\{\beta\langle l(x_t, z_t; w_t)\rangle - \eta_t\},$$

where $\dot{}$ denotes $d/dt$ and $\langle\ \rangle$ the average over the current $(x, z)$, we have

$$\dot{e}_t = -2\eta_t e_t, \tag{19}$$
$$\dot{\eta}_t = \alpha\beta\eta_t e_t - \alpha\eta_t^2. \tag{20}$$

The behavior of the above equation is interesting : The origin $(0,0)$ is its attractor. However, the basin of attraction has a fractal boundary. Anyway, starting from an adequate initial value, it has the solution of the form

$$e_t \approx \frac{1}{\beta} \left( \frac{1}{2} - \frac{1}{\alpha} \right) \frac{1}{t}, \tag{21}$$

$$\eta_t \approx \frac{1}{2t}. \tag{22}$$

This proves the $1/t$ convergence rate of the generalization error, that is optimal in order for any estimator $\hat{w}_t$ converging to $w^*$.

## 6  Riemannian structures of simple perceptrons

We first study the parameter space $S$ of simple perceptrons to obtain an explicit form of the metric $G$ and its inverse $G^{-1}$. This suggests how to calculate the metric in the parameter space of multilayer perceptrons.

Let us consider a simple perceptron with input $x$ and output $z$. Let $w$ be its connection weight vector. For the analog stochastic perceptron, its input-output behavior is described by $z = f(w'x) + n$, where $n$ denotes a random noise subject to the normal distribution $N(0, \sigma^2)$ and $f$ is the hyperbolic tangent,

$$f(u) = \frac{1 - e^{-u}}{1 + e^{-u}}.$$

In order to calculate the metric $G$ explicitly, let $e_w$ be the unit column vector in the direction of $w$ in the Euclidean space $\boldsymbol{R}^n$,

$$e_w = \frac{w}{\| w \|},$$

where $\| w \|$ is the Euclidean norm. We then have the following theorem.

**Theorem 2.** The Fisher metric $G$ and its inverse $G^{-1}$ are given by

$$G(w) = c_1(w)I + \{c_2(w) - c_1(w)\} e_w e_w', \tag{23}$$

$$G^{-1}(w) = \frac{1}{c_1(w)} I + \left( \frac{1}{c_2(w)} - \frac{1}{c_1(w)} \right) e_w e_w'. \tag{24}$$

where $w = |w|$ (Euclidean norm) and $c_1(w)$ and $c_2(w)$ are given by

$$c_1(w) = \frac{1}{4\sqrt{2\pi}\sigma^2} \int \{f^2(w\varepsilon) - 1\}^2 \exp\left\{-\frac{1}{2}\varepsilon^2\right\} d\varepsilon, \tag{25}$$

$$c_2(w) = \frac{1}{4\sqrt{2\pi}\sigma^2} \int \{f^2(w\varepsilon) - 1\}^2 \varepsilon^2 \exp\left\{-\frac{1}{2}\varepsilon^2\right\} d\varepsilon. \tag{26}$$

**Theorem 3.** The Jeffrey prior is given by

$$\sqrt{|G(w)|} = \frac{1}{V_n} \sqrt{c_2(w)\{c_1(w)\}^{n-1}}. \tag{27}$$

## 7  The natural gradient for blind separation of mixtured signals

Let $s = (s_1, \cdots, s_n)$ be $n$ source signals which are $n$ independent random variables. We assume that their $n$ mixtures $x = (x_1, \cdots, x_n)$,

$$x = As \tag{28}$$

are observed. Here, $A$ is a matrix. When $s$ is time serieses, we observe $\boldsymbol{x}(1), \cdots, \boldsymbol{x}(t)$. The problem of blind separation is to estimate $W = A^{-1}$ adaptively from $\boldsymbol{x}(t)$, $t = 1, 2, 3, \cdots$ without knowing $\boldsymbol{s}(t)$ nor $A$. We can then recover original $\boldsymbol{s}$ by

$$\boldsymbol{y} = \hat{W}\boldsymbol{x} \tag{29}$$

when $\hat{W} = A^{-1}$. Let $W \in Gl(n)$, that is a nonsingular $n \times n$-matrix, and $\phi(W)$ be a scalar function. This is given by a measure of independence such as $\phi(W) = KL[\tilde{p}(\boldsymbol{y}); p(\boldsymbol{y})]$, which is represented by the expectation of a loss function. We define the natural gradient of $\phi(W)$.

Now we return to our manifold $Gl(n)$ of matrices. It has the Lie group structure : Any $A \in Gl(n)$ maps $Gl(n)$ to $Gl(n)$ by $W \to WA$. We impose that the Riemannian structure should be invariant by this operation $A$.

We can then prove that the natural gradient in this case is

$$\tilde{\nabla}\phi = \nabla\phi W'W. \tag{30}$$

The natural gradient works surprisingly well for adaptive blind signal separation Amari et al. [1995], Cardoso and Laheld [1996].

# References

[1] S. Amari. Theory of adaptive pattern classifiers, *IEEE Trans.*, **EC-16**, No.3, 299–307, 1967.

[2] S. Amari. *Differential-Geometrical Methods in Statistics, Lecture Notes in Statistics*, vol.28, Springer, 1985.

[3] S. Amari. Information geometry of the EM and em algorithms for neural networks, *Neural Networks*, **8**, No.9, 1379–1408, 1995.

[4] S. Amari, A. Cichocki and H.H. Yang. A new learning algorithm for blind signal separation, in *NIPS'95*, vol.8, 1996, MIT Press, Cambridge, Mass.

[5] S. Amari, K. Kurata, H. Nagaoka. Information geometry of Boltzmann machines, *IEEE Trans. on Neural Networks*, **3**, 260–271, 1992.

[6] J. F. Cardoso and Beate Laheld. Equivariant adaptive source separation, to appear *IEEE Trans. on Signal Processing*, 1996.

[7] T. M. Heskes and B. Kappen. Learning processes in neural networks, *Physical Review A*, **440**, 2718–2726, 1991.

[8] D. Rumelhart, G.E. Hinton and R. J. Williams. Learning internal representation, in *Parallel Distributed Processing: Explorations in the Microstructure of Cognition*, **1**, *Foundations*, MIT Press, Cambridge, MA, 1986.

[9] H. Sompolinsky, N. Barkai and H. S. Seung. On-line learning of dichotomies: algorithms and learning curves, *Neural Networks: The statistical Mechanics Perspective*, Proceedings of the CTP-PBSRI Joint Workshop on Theoretical Physics, J.-H. Oh et al eds, 105–130, 1995.